# Input Reconstruction Reliability Estimation

**Dean A. Pomerleau**
School of Computer Science
Carnegie Mellon University
Pittsburgh, PA 15213

## Abstract

This paper describes a technique called *Input Reconstruction Reliability Estimation* (IRRE) for determining the response reliability of a restricted class of multi-layer perceptrons (MLPs). The technique uses a network's ability to accurately encode the input pattern in its internal representation as a measure of its reliability. The more accurately a network is able to reconstruct the input pattern from its internal representation, the more reliable the network is considered to be. IRRE is provides a good estimate of the reliability of MLPs trained for autonomous driving. Results are presented in which the reliability estimates provided by IRRE are used to select between networks trained for different driving situations.

## 1 Introduction

In many real world domains it is important to know the reliability of a network's response since a single network cannot be expected to accurately handle all the possible inputs. Ideally, a network should not only provide a response to a given input pattern, but also an indication of the likelihood that its response is "correct". This reliability measure could be used to weight the outputs from multiple networks and to determine when a new network needs to be trained.

This paper describes a technique for estimating a network's reliability called *Input Reconstruction Reliability Estimation* (IRRE). IRRE relies on the fact that the hidden representation developed by an artificial neural network can be considered to be a compressed representation of important input features. For example, when the network shown in Figure 1 is trained to produce the correct steering direction from images of the road ahead, the hidden units learn to encode the position and orientation of important features like the road edges and lane markers (See [Pomerleau, 1991] for more details). Because there are many fewer hidden units than input units in the network, the hidden units cannot accurately represent all the details of an

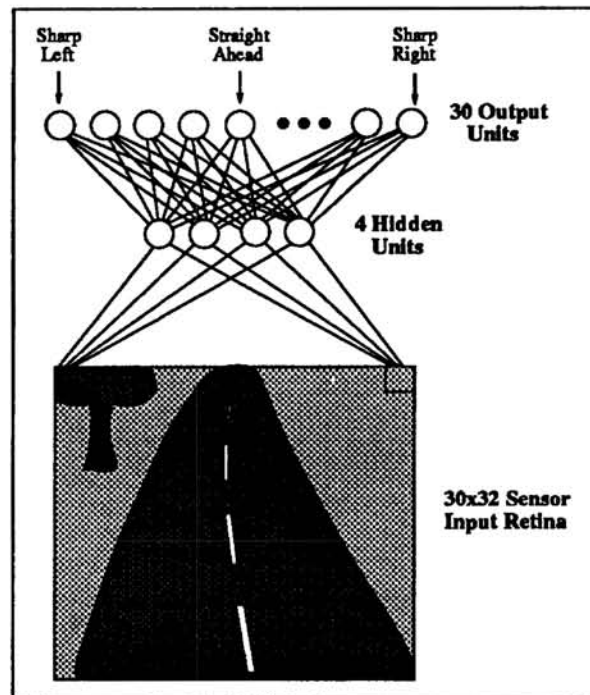

Figure 1: Original driving network architecture.

arbitrary input pattern. Instead, the hidden units learn to devote their limited representational capabilities to encoding the position and orientation of consistent, frequently-occurring features from the training set. When presented with an atypical input, such as a road with a different number of lanes, the feature detectors developed by the hidden units will not be capable of accurately encode all the actual input features.

Input Reconstruction Reliability Estimation exploits this limitation in representational capacity to estimate a network's reliability. In IRRE, the network's internal representation is used to reconstruct in the input pattern being presented. The more closely the reconstructed input matches the actual input, the more familiar the input and hence the more reliable the network's response.

## 2   Reconstructing the Input

IRRE utilized an additional set of output units to perform input reconstruction called the encoder output array, as depicted in Figure 2. This second set of output units has the same dimensionality as the input retina. In the experiments described in this paper, the input layer and encoder output array have 30 rows and 32 columns. The desired activation for each of these additional output units is identical to the activation of the corresponding input unit. In essence, these additional output units turn the network into an autoencoder.

The network is trained using backpropagation both to produce the correct steering response on the steering output units, and to reconstruct the input image as accurately as possible on the encoder output array. During the training process, the network is presented with several hundred images taken with a camera onboard our test vehicle as a person drives (See [Pomerleau, 1991] for more details). Training typically requires approximately 3 minutes during which the person drives over a 1/4 to 1/2 mile stretch of road.

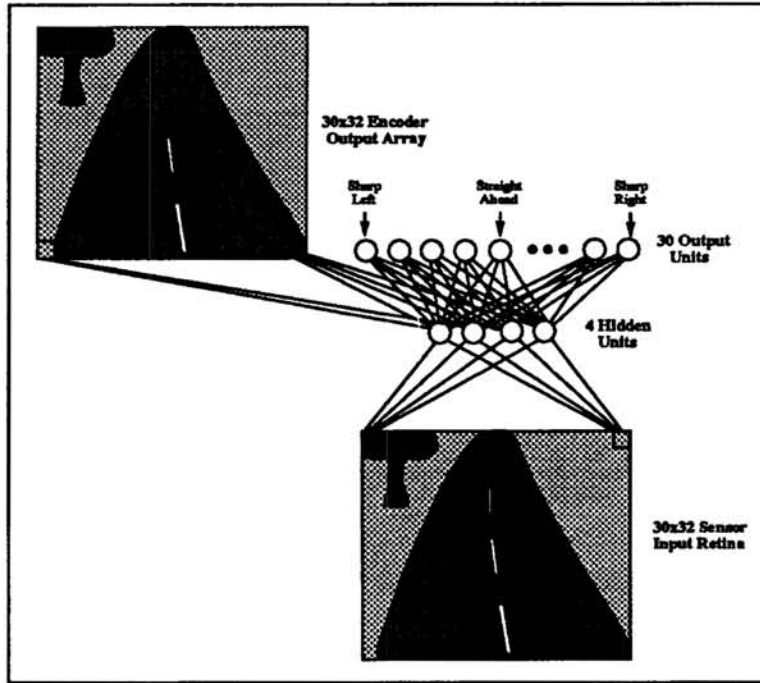

Figure 2: Network architecture augmented to include an encoder output array.

During testing on a new stretch of road, images are presented to the network and activation is propagated forward through the network to produce a steering response and a reconstructed input image. The reliability of the steering response is estimated by computing the correlation coefficient $\rho(\mathbf{I}, \mathbf{R})$ between the activation levels of units in the actual input image $\mathbf{I}$ and the reconstructed input image $\mathbf{R}$ using the following formula:

$$\rho(\mathbf{I}, \mathbf{R}) = \frac{\overline{\mathbf{IR}} - \overline{\mathbf{I}} \cdot \overline{\mathbf{R}}}{\sigma_I \, \sigma_R}$$

where $\overline{\mathbf{I}}$ and $\overline{\mathbf{R}}$ are the mean activation value of the actual and the reconstructed images, $\overline{\mathbf{IR}}$ is the mean of the set formed by the unit-wise product of the two images, and $\sigma_I$ and $\sigma_R$ represent the standard deviations of the activation values of each image. The higher the correlation between the two images, the more reliable the network's response is estimated to be. The reason correlation is used to measure the degrees of match between the two images is that, unlike Euclidean distance, the correlation measure is invariant to differences in the mean and variance between the two images. This is important since the mean and variance of the input and the reconstructed images can sometimes vary, even when the input image depicts a familiar situation.

## 3   Results and Applications

The degree of correlation between the actual and the reconstructed input images is an extremely good indicator of network response accuracy in the domain of autonomous driving, as shown in Figure 3. It shows a trained network's steering error and reconstruction error as the vehicle drives down a quarter mile stretch of road that starts out as a single lane path and eventually becomes a two-lane street. The solid line indicates the network's steering error, as measured by the difference in turn curvature between the network's steering response and a person's

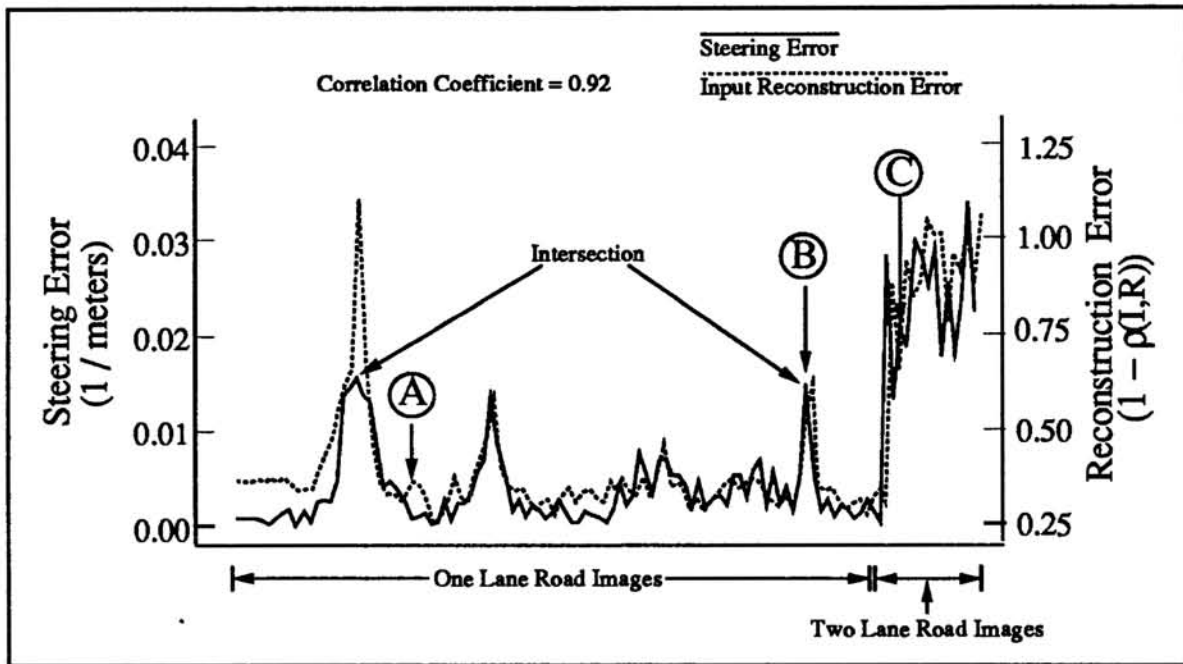

Figure 3: Reconstruction error obtained using autoencoder reconstruction versus network steering error over a stretch of one-lane and two-lane road.

steering response at that point along the road. The dashed line represents the network's "reconstruction error", which is defined to be the degree of statical independence between the actual and reconstructed images, or $1 - \rho(\mathbf{I}, \mathbf{R})$.

The two curves are nearly identical, having a correlation coefficient of 0.92. This close match between the curves demonstrates that when the network is unable to accurately reconstruct the input image, it is also probably suggesting an incorrect steering direction. Visual inspection of the actual and reconstructed input images demonstrates that the degree of resemblance between them is a good indication of the actual input's familiarity, as shown in Figure 4. It depicts the input image, network response, and reconstructed input at the three points along the road, labeled A, B and C in Figure 3. When presented with the image at point A, which closely resembles patterns from training set, the network's reconstructed image closely resembles the actual input, as shown by the close correspondence between the images labeled "Input Acts" and "Reconstructed Input" in the left column of Figure 4. This close correspondence between the input and reconstructed images suggests that the network can reliably steer in this situation. It in fact it can steer accurately on this image, as demonstrated by the close match between the network's steering response labeled "Output Acts" and the desired steering response labeled "Target Acts" in the upper left corner of Figure 4.

When presented with a situation the network did not encounter during training, such as the fork image and the two-lane road image shown in the other two columns of Figure 4, the reconstructed image bears much less resemblance to the original input. This suggests that the network is confused. This confusion results in an incorrect steering response, illustrated in the discrepancy between the network's steering response and the target steering response for the two atypical images.

The reliability prediction provided by IRRE has been used to improve the performance of the neural network based autonomous driving system in a number of ways. The simplest is

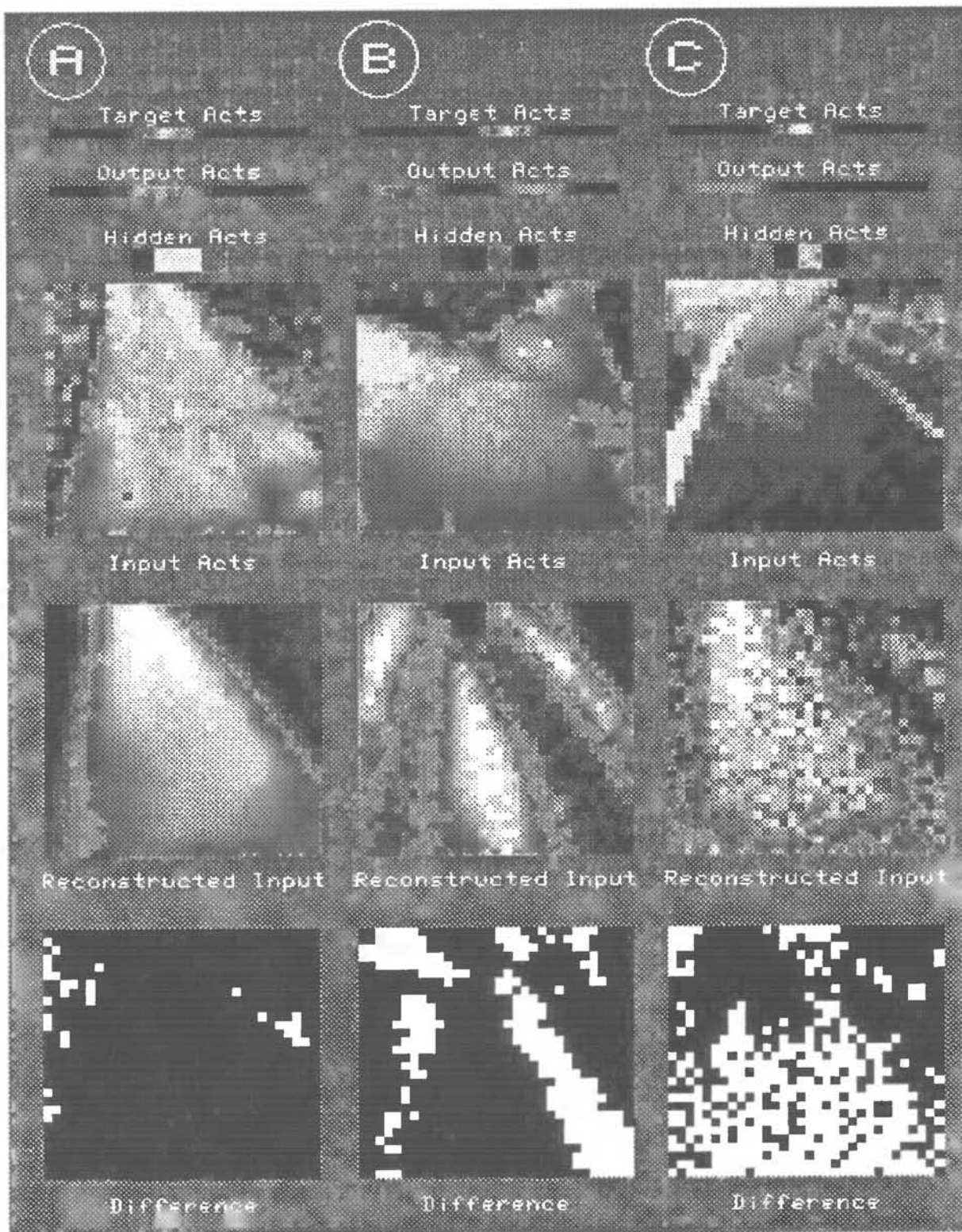

Figure 4: The actual input, the reconstructed input and the point-wise absolute difference between them on a road image similar to those in the training set (labeled A), and on two atypical images (labeled B and C).

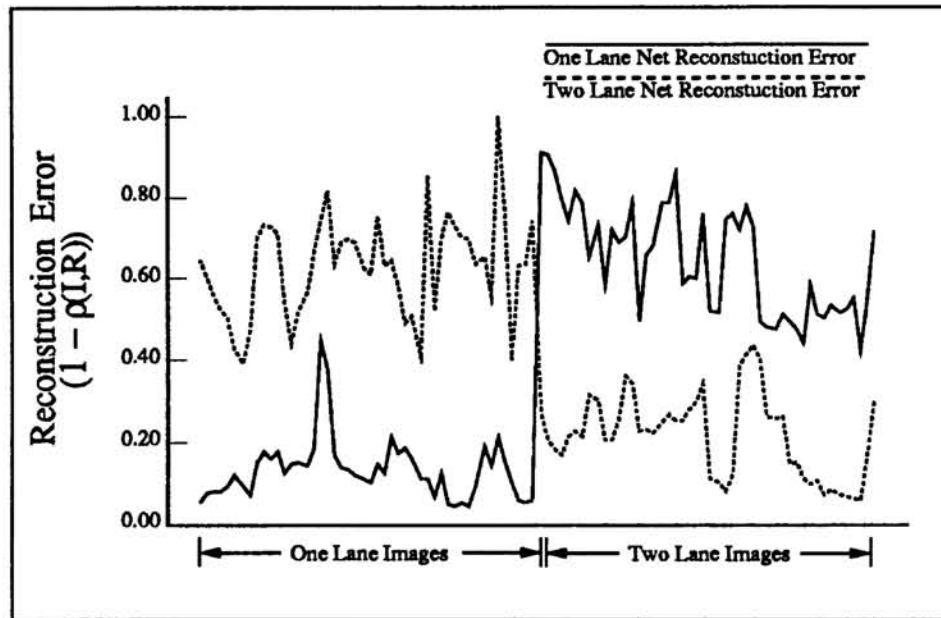

Figure 5: Reconstruction error of networks trained for one-lane road driving (solid line) and two-lane road driving (dashed line).

to use IRRE to control vehicle speed. The more accurate the input reconstruction, the more confident the network, and hence the faster the system drives the vehicle. A second use the system makes of the reliability estimate provided by IRRE is to update the vehicle's position on a rough map of the area being driven over. When the map indicates there should be an intersection or other confusing situation up ahead, a subsequent sharp rise in reconstruction error is a good indication that the vehicle has actually reached that location, allowing the system to pinpoint the vehicle's position. Knowing the vehicle's location on a map is useful for integrating symbolic processing, such as planning, into the neural network driving system (for more details, see [Pomerleau et al., 1991]).

Figure 5 illustrates how IRRE can be used to integrate the outputs from multiple expert networks. The two lines in this graph illustrate the reconstruction error of two networks, one trained to steer on one-lane roads (solid line), and the other trained to steer on two-lane roads (dashed line). The reconstruction error for the one-lane network is low on the one-lane road images, and high on the two-lane road images. The opposite is true for the network trained for two-lane road driving. The reliability estimate provided by IRRE allows the system to determine which network is most reliable for driving in the current situation. By simulating multiple networks in parallel, and then selecting the one with the highest reliability, the system have been able to drive the Navlab test vehicle on one, two and four lane roads automatically at speeds of up to 30 miles per hour.

## 4   Discussion

The effectiveness of input reconstruction reliable estimation stems from the fact that the network has a small number of hidden units and is only trained in a narrow range of situations. These constraints prevent the network from faithfully encoding arbitrary input patterns. Instead, the hidden units learn to encode features in the training images that are most important for the task. Baldi and Hornik [Baldi & Hornik, 1989] have shown that if an autoencoder

network with a single layer of $N$ linear hidden units is trained with back-propagation, the activation levels of the hidden units will represent the first $N$ principal components of the training set. Since the units in the driving network are non-linear, this assertion does not strictly hold in this case. However, Cottrell and Munro [Cottrell & Munro, 1988] have found empirically that autoencoder networks with a sigmoidal activation function develop hidden units that span the principal subspace of the training images, with some noise on the first principal component due to network non-linearity. Because the principal components represent the dimensions along which the training examples varies most, it can be shown that using linear combinations of the principal components to represent the individual training patterns optimally preserves the information contained in the training set [Linsker, 1989].

However the compressed representation developed by a linear autoencoder network is only optimal for encoding images from the same distribution as the training set. When presented with images very different from those in the training set, the image reconstructed from the internal representation is not as accurate. The results presented in this paper demonstrate that this reconstruction error can be employed to estimate the likelihood and magnitude of error in MLPs trained for autonomous driving.

However the input reconstruction technique presented here has a serious potential shortcoming, namely that it forces the network's hidden units to encode all input features, including potentially irrelevant ones. While this increased representation load on the hidden units has the potential to degrade network performance, this effect has not been observed in the tests conducted so far. In support of this finding, Gluck [Gluck, personal communications] has found that forcing a network to autoencode its input frequently *improves* its generalization. In [Gluck & Myers, 1992], Gluck and Myers use the representation developed by an autoencoder network as a model for simple types of learning in biological systems. The model suggests that the hippocampus acts as an autoencoder, developing internal representations that are then used to perform other tasks.

But if interference from the autoencoder task proves to be a problem, one way to eliminate it would be to have separate groups of hidden units connected exclusively to one group of outputs or the other. Having a separate set of hidden units for the autoencoder task would ensure that the representation developed for the input reconstruction does not interfere with representation developed for the "normal" task. It remains to be seen if this decoupling of internal representations will adversely affect IRRE's ability to predict network errors.

As a technique for multi-network integration, IRRE has several advantages over existing connectionist arbitration methods, such as Hampshire and Waibel's Meta-Pi architecture [Hampshire & Waibel, 1992] and the Adaptive Mixture of Experts Model of Jacobs et al. [Jacobs et al., 1991]. It is a more modular approach, since each expert can be trained entirely in isolation and then later combined with other experts without any additional training by simply selecting the most reliable network for the current input. Since IRRE provides an absolute measure of a single network's reliability, and not just a measure of how appropriate a network is relative to others, IRRE can be also used to determine when none of the experts is capable of coping with the current situation.

A potentially interesting extension to IRRE is the development of techniques for reasoning about the difference between the actual input and the reconstructed input. For instance, it should be possible to recognize when the vehicle has reached a fork in the road by the characteristic mistakes the network makes in reconstructing the input image. Another important component of future work in is to test the ability of IRRE to estimate network reliability in domains other than autonomous driving.

## Acknowledgements

I thank Dave Touretzky, Chuck Thorpe and the entire Unmanned Ground Vehicle Group at CMU for their support and suggestions. Principle support for this research has come from DARPA, under contracts "Perception for Outdoor Navigation" (contract number DACA76-89-C-0014, monitored by the US Army Topographic Engineering Center) and "Unmanned Ground Vehicle System" (contract number DAAE07-90-C-R059, monitored by TACOM).

## References

[Baldi & Hornik, 1989] Baldi, P. and Hornik, K. (1989) Neural networks and principal component analysis: Learning from examples without local minima. *Neural Networks, Vol. 2* pp. 53-58.

[Cottrell & Munro, 1988] Cottrell, G.W. and Munro, P. (1988) Principal components analysis of images via back-propagation. *Proc. Soc. of Photo-Optical Instr. Eng.*, Cambridge MA.

[Gluck, personal communications] Gluck, M.A. (1992) Personal Communications. Rutgers Univ., Newark NJ.

[Gluck & Myers, 1992] Gluck, M.A. and Myers, C.E. (1992) Hippocampal function in representation and generalization: a computational theory. *Proc. 1992 Cogn. Sci. Soc. Conf.* Hillsdale, NJ: Erlbaum Assoc.

[Hampshire & Waibel, 1992] Hampshire J.B. and Waibel, A.H. (1989) The Meta-Pi network: building distributed knowledge representations for robust pattern recognition. *IEEE Trans. on Pattern Analysis and Machine Intelligence.*

[Jacobs et al., 1991] Jacobs, R.A., Jordan, M.I. Nowlan, S.J. and Hinton, G.E. (1991) Adaptive mixtures of local experts. *Neural Computation, 3:1*, Terrence Sejnowski (ed).

[Linsker, 1989] Linsker, R. (1989) Designing a sensory processing system: What can be learned from principal component analysis? IBM Technical Report RC14983 (#66896).

[Pomerleau, 1991] Pomerleau, D.A. (1991) Efficient Training of Artificial Neural Networks for Autonomous Navigation. *Neural Computation 3:1*, Terrence Sejnowski (ed).

[Pomerleau et al., 1991] Pomerleau, D.A., Gowdy, J., Thorpe, C.E. (1991) Combining artificial neural networks and symbolic processing for autonomous robot guidance. *Engineering Applications of Artificial Intelligence, 4:4* pp. 279-285.
